# A LARGE-SCALE NEURAL NETWORK WHICH RECOGNIZES HANDWRITTEN KANJI CHARACTERS

Yoshihiro Mori        Kazuki Joe
ATR Auditory and Visual Perception Research Laboratories
Sanpeidani Inuidani Seika-cho Soraku-gun Kyoto 619-02 Japan

## ABSTRACT

We propose a new way to construct a large-scale neural network for 3,000 handwritten Kanji characters recognition. This neural network consists of 3 parts: a collection of small-scale networks which are trained individually on a small number of Kanji characters; a network which integrates the output from the small-scale networks, and a process to facilitate the integration of these neworks. The recognition rate of the total system is comparable with those of the small-scale networks. Our results indicate that the proposed method is effective for constructing a large-scale network without loss of recognition performance.

## 1 INTRODUCTION

Neural networks have been applied to recognition tasks in many fields, with good results [Denker,1988][Mori,1988][Weideman,1989]. They have performed better than conventional methods. However these networks currently operate with only a few categories, about 20 to 30. The Japanese writing system at present is composed of about 3,000 characters. For a network to recognize this many characters, it must be given a large number of categories while maintaining its level of performance.

To train small-scale neural networks is not a difficult task. Therefore, exploring methods for integrating these small-scale neural networks is important to construct a large-scale network. If such methods could integrate small-scale networks without loss of the performance, the scale of neural networks would be extended dramatically. In this paper, we propose such a method for constructing a large-scale network whose object is to recognize 3,000 handwritten Kanji characters, and report the result of a part of this network. This method is not limited to systems for character recognition, and can be applied to any system which recognizes many categories.

## 2 STRATEGIES FOR A LARGE-SCALE NETWORK

Knowing the current recognition and generalization capacity of a neural network, we realized that constructing a large-scale monolithic network would not be efficient or

effective. Instead, from the start we decided on a building blocks approach [Mori,1988][Waibel,1988]. There are two strategies to mix many small-scale networks.

## 2.1 Selective Neural Network (SNN)

In this strategy, a large-scale neural network is made from many small-scale networks which are trained individually on a small number of categories, and a network (SNN) which selects the appropriate small-scale network (Fig. 1). The advantage of this strategy is that the information passed to a selected small-scale networks is always appropriate for that network. Therefore, training these small-scale networks is very easy. But on the other hand, increasing the number of categories will substantially increase the training time of the SNN, and may make it harder for the SNN to retain high performance. Furthermore, the error rate of the SNN will limit the performance of the whole system.

## 2.2 Integrative Neural Network (INN)

In this strategy, a large-scale neural network is made from many small-scale networks which are trained individually on a small number of categories, and a network (INN) which integrates the output from these small-scale networks(Fig. 2). The advantage of this strategy is that every small-scale network gets information and contributes to finding the right answer. Therefore, it is possible to use the knowledge distributed among each small-scale network. But in some respects, various devices are needed to make the integration easier.

The common advantage with both strategies just mentioned is that the size of each neural network is relatively small, and it does not take a long time to train these networks. Each small-scale networks is considered an independent part of the whole system. Therefore, retraining these networks (to improve the performance of the whole system) will not take too long.

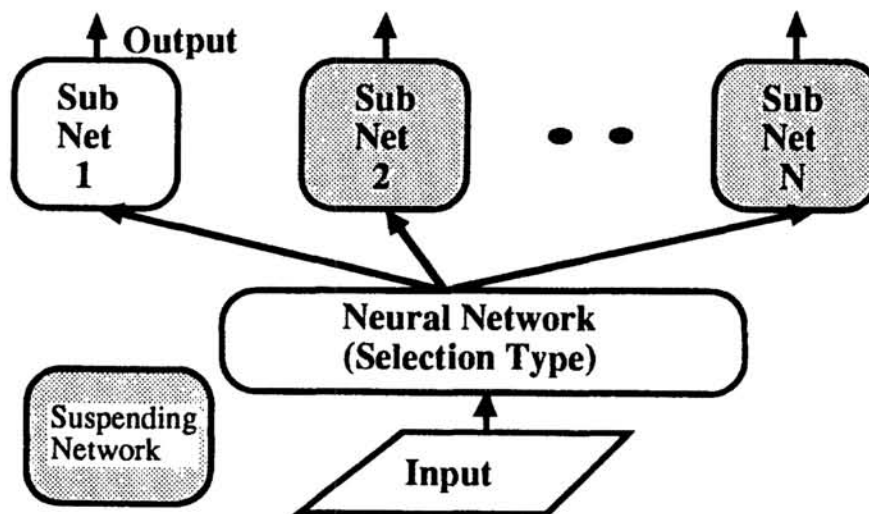

Fig. 1 SNN Strategy

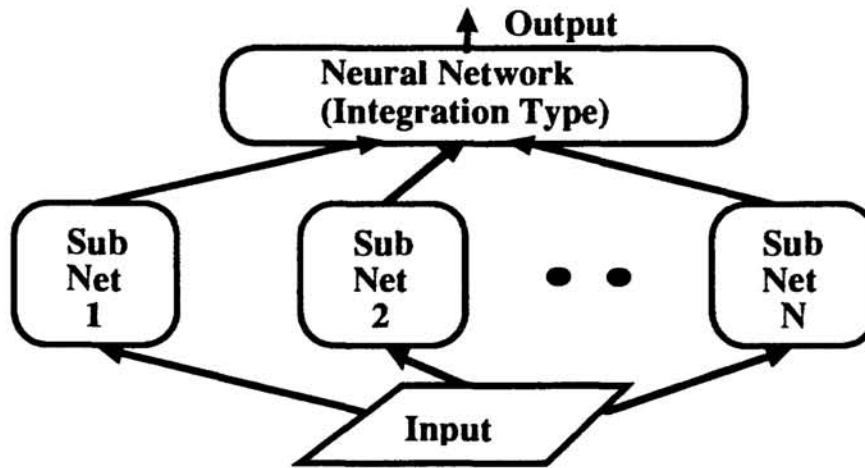

Fig. 2  INN Strategy

# 3 STRUCTURE OF LARGE-SCALE NETWORK

The whole system is constructed using three kinds of neural networks. The first one, called a SubNet, is an ordinary three layered feed forward type neural network trained using the Back Propagation learning algorithm. The second kind of network is called a SuperNet. This neural network makes its decision by integrating the outputs from all the SubNets. This network is also a 3-layered feed-forward net, but is larger than the Subnets. The last network, which we call an OtherFilter, is devised to improve the integration of the SuperNet. This OtherFilter network was designed using the LVQ algorithm [Khonen,1988]. There are also some changes made in the BP learning algorithm especially for pattern recognition [Joe,1989].

We decided that, based on the time it takes for learning, there should be 9 categories in each small-scale network. The 3,000 characters are separated into these small groups through the K-means clustering method, which allows similar characters to be grouped together. The separation occurs in two stages. First, 11 groups of 270 characters each are formed, then each group is separated into 30 smaller units. In this way, 330 groups of 9 characters each are obtained. We choose the INN strategy to use distributed knowledge to full advantage. The 9-character units are SubNets, which are integrated in 2 stages. First 30 SubNets are integrated by a higher level network SuperNet. Altogether, 11 SuperNets are needed to recognize all 3,000 characters. SuperNets are in turn integrated by a higher level network, the HyperNet. More precisely, the role and structure of these kinds of networks are as follows:

## 3.1  SubNet

A feature vector extracted from handwritten patterns is used as the input (described in Section 4.1). The number of units in the output layer is the same as the number of categories to be recognized by the SubNet. In short, the role of a SubNet is to output the similarity between the input pattern and the categories allotted to the SubNet. (Fig. 3)

## 3.2  SuperNet

The outputs from each SubNet filtered by the OtherFilter network are used as the input to

the SuperNet. The number of units in an output layer is the same as the number of SubNets belonging to a SuperNet. In short, the role of SuperNet is to select the SubNet which covers the category corresponding to the input patterns. (Fig. 5)

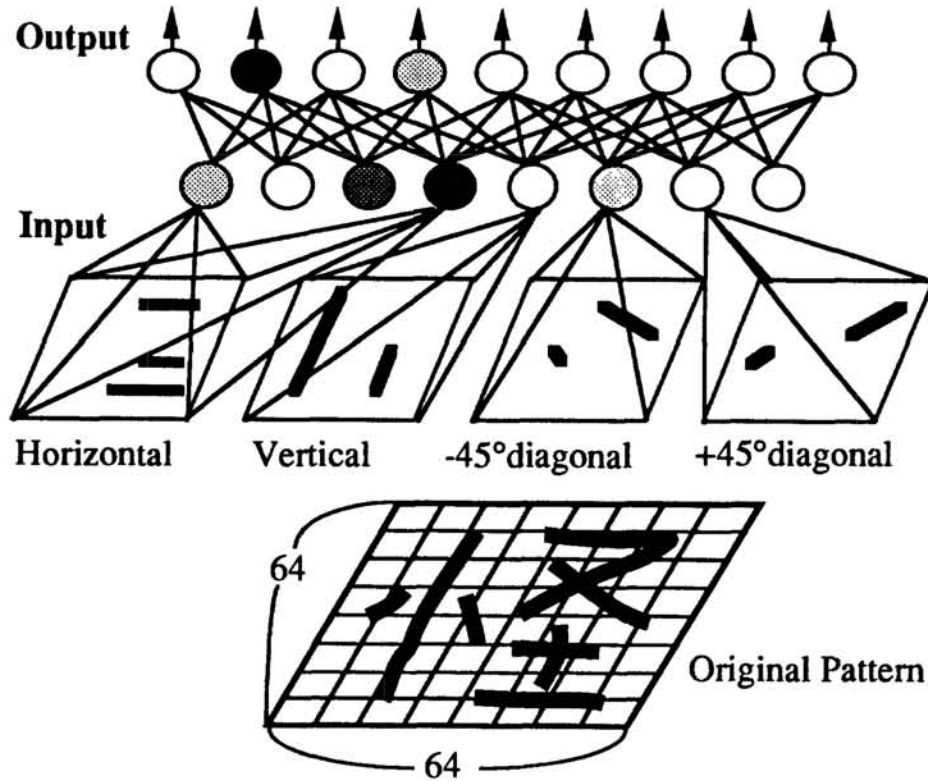

Fig. 3 SubNet

### 3.3 OtherFIlter

45(9x5) reference vectors are assigned to each SubNet. LVQ is used to adapt these reference vectors, so that each input vector has a reference of the correct SubNet as its closest reference vector. The OtherFilter method is to first measure the distance between all the reference vectors and one input vector. The mean distance and normal deviation of distance are calculated. The distance between a SubNet and an input vector is defined to be the smallest distance of that SubNet's reference vectors to the input vector.

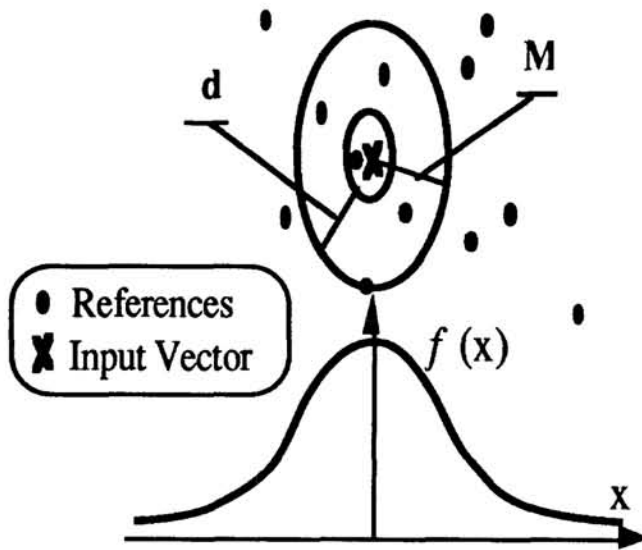

Fig4. Shape of OtherFilter

$$f(x_n)=1 / (1+ e^{(x_n-M+2d)/Cd}) \quad (1)$$

$x_n$  : The Distance of Nth SubNet
$M$   : The Mean of $x_n$
$d$   : The Variance of $x_n$
$C$   : Constant

This distance modified by equation (1) is multiplied by the outputs of the SubNet, and fed into the SuperNet. The outputs of SubNets whose distance is greater than the mean distance are suppressed, and the outputs of SubNets whose distance is smaller than the mean distance are amplified. In this way, the outputs of SubNets are modified to improve the integration of the higher level SuperNet. (Fig. 5)

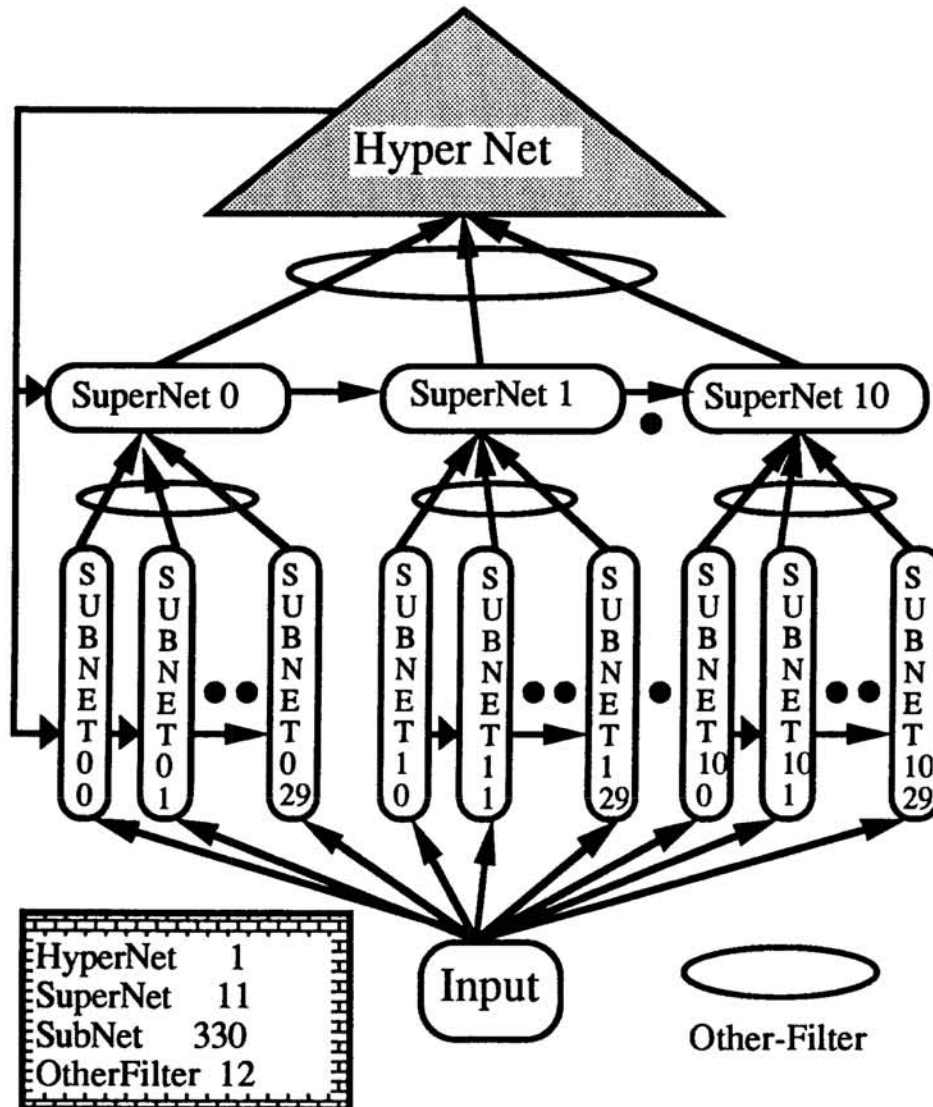

Fig5. Outline of the Whole System

# 4 RECOGNITION EXPERIMENT

## 4.1 TRAINING PATTERN

The training samples for this network were chosen from a database of about 3000 Kanji characters [Saito 1985]. For each character, there are 200 handwritten samples from different writers. 100 are used as training samples, and the remaining 100 are used to test recognition accuracy of the trained network. All samples in the database consist of 64 by 63 dots.

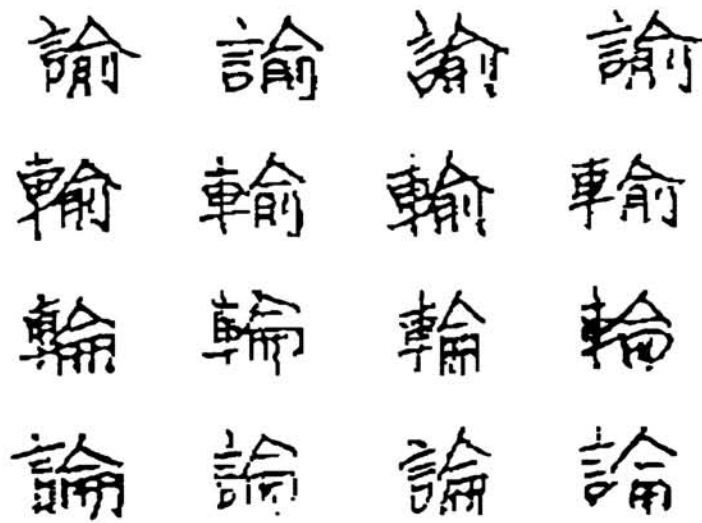

Fig. 6  Examples of training pattern

## 4.2 LDCD FEATURE

If we were to use this pattern as the input to our neural net, the number of units required in the input layer would be too large for the computational abilities of current computers. Therefore, a feature vector extracted from the handwritten patterns is used as the input. In the "LDCD feature" [Hagita 1983], there are 256 dimensions computing a line segment length along four directions: horizontal, vertical, and two diagonals in the 8 by 8 squares into which the handwritten samples are divided.

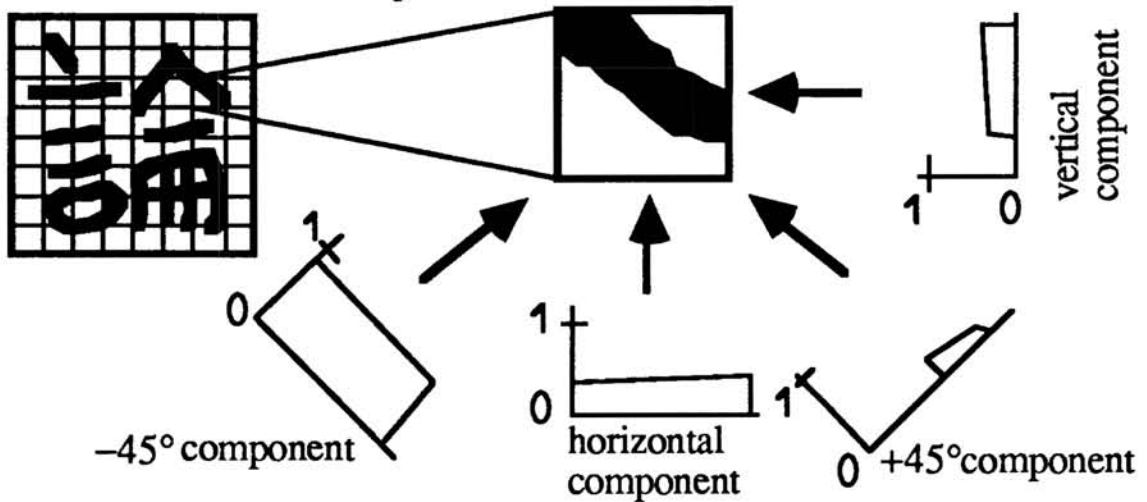

Fig 7.  LDCD Feature

## 4.3 RECOGNITION RESULTS

In the work reported here, one SuperNet, 30 SubNets and one OtherFilter were constructed for recognition experiments. SubNets were trained until the recognition of training samples reaches at least 99%. With these SubNets, the mean recognition rate of test patterns was 92%. This recognition rate is higher than that of conventional methods. A SuperNet which integrates the output modified by OtherFilter from 30 trained SubNets

was then constructed. The number of units in the input layer of the SuperNet was 270. This SuperNet was trained until the performance of training samples becomes at least 93%. With this SuperNet, the recognition rate of test patterns was 74%, though that of OtherFilter was 72%. The recognition rate of a system without the OtherFilter of test patterns was 55%.

# 5  CONCLUSION

We have here proposed a new way of constructing a large-scale neural network for the recognition of 3,000 handwritten Kanji characters. With this method, a system recognizing 270 Kanji characters was constructed. This system will become a part of a system recognizing 3,000 Kanji characters. Only a modest training time was necessary owing to the modular nature of the system. Moreover, this modularity means that only a modest re-training time is necessary for retraining an erroneous neural network in the whole system. The overall system performance can be improved by retraining just that neural network, and there is no need to retrain the whole system. However, the performance of the OtherFilter is not satisfactory. We intend to improve the OtherFilter, and build a large-scale network for the recognition of 3,000 handwritten Kanji characters by the method reported here.

## Acknowledgments

We are grateful to Dr. Yodogawa for his support and encouragement. Special thanks to Dr. Sei Miyake for the ideas he provided in our many discussions. The authors would like to acknowledge, with thanks, the help of Erik McDermott for his valuable assistance in writing this paper in English.

## References

[Denier,1988]    J.S.Denker, W.R.Gardner, H.P. Graf, D.Henderson, R.E. Howard, W.Hubbard, L.D.Jackel, H.S.Baird, I.Guyon : "Neural Network Recognizer for Hand-Written ZIP Code Digits", NEURAL INFORMATION PROCESSING SYSTEMS 1, pp.323-331, Morgan Kaufmann, 1988
[Mori,1988]    Y.Mori, K.Yokosawa : "Neural Networks that Learn to Discriminate Similar Kanji Characters", NEURAL INFORMATION PROCESSING SYSTEMS 1, pp.332-339, Morgan Kaufmann, 1988
[Weideman,1989]W.E.Weideman, M.T.Manry, H.C.Yau ; "A COMPARISON OF A NEAREST NEIGHBOR CLASSIFIER AND A NEURAL NETWORK FOR NUMERIC HANDPRINT CHARACTER RECOGNITION", IJCNN89(Washington), Vol.I, pp.117-120, June 1989

[Waibel,1988]    Alex Waibel, "Consonant Recognition by Modular Construction of Large Phonemic Time-Delay Neural Networks", NEURAL INFORMATION PROCESSING SYSTEMS 1, pp.215-223, Morgan Kaufmann, 1988

[Joe,1989]    K.Joe, Y.Mori, S.Miyake : "Simulation of a Large-Scale Neural Networks on a Parallel Computer", 4th Hypercube Concurrent Computers,1989

[Khonen,1988]    T.Kohonen, G.Barna, R.Chrisley : "Statistical Pattern Recognition with Neural Networks", IEEE, Proc.of ICNN, Vol.I, pp.61-68, July 1988

[Saito,1985]    T.Saito, H.Yamada, K.Yamamoto : "On the Data Base ETL9 of Handprinted Characters in JIS Chinese Characters and Its Analysis", J68-D, 4, 757-764, 1985

[Hagita,1983]    N.Hagita, S.Naito, I.Masuda : "Recognition of Handprinted Chinese Characters by Global and Local Direction Contributivity Density-Feature", J66-D, 6, 722-729,1983
